# SIMULATIONS SUGGEST INFORMATION PROCESSING ROLES FOR THE DIVERSE CURRENTS IN HIPPOCAMPAL NEURONS

Lyle J. Borg-Graham
Harvard-MIT Division of Health Sciences and Technology and
Center for Biological Information Processing,
Massachusetts Institute of Technology, Cambridge, Massachusetts 02139

## ABSTRACT

A computer model of the hippocampal pyramidal cell (HPC) is described which integrates data from a variety of sources in order to develop a consistent description for this cell type. The model presently includes descriptions of eleven non-linear somatic currents of the HPC, and the electrotonic structure of the neuron is modelled with a soma/short-cable approximation. Model simulations qualitatively or quantitatively reproduce a wide range of somatic electrical behavior in HPCs, and demonstrate possible roles for the various currents in information processing.

## 1 The Computational Properties of Neurons

There are several substrates for neuronal computation, including connectivity, synapses, morphometrics of dendritic trees, linear parameters of cell membrane, as well as non-linear, time-varying membrane conductances, also referred to as currents or channels. In the classical description of neuronal function, the contribution of membrane channels is constrained to that of generating the action potential, setting firing threshold, and establishing the relationship between (steady-state) stimulus intensity and firing frequency. However, it is becoming clear that the role of these channels may be much more complex, resulting in a variety of novel "computational operators" that reflect the information processing occurring in the biological neural net.

## 2 Modelling Hippocampal Neurons

Over the past decade a wide variety of non-linear ion channels, have been described for many excitable cells, in particular several kinds of neurons. One such neuron is the hippocampal pyramidal cell (HPC). HPC channels are marked by their wide range of temporal, voltage-dependent, and chemical-dependent characteristics, which results in very complex behavior or responses of these stereotypical cortical integrating cells. For example, some HPC channels are activated (opened) transiently and quickly, thus primarily affecting the action potential shape. Other channels have longer kinetics, modulating the response of HPCs over hundreds of milliseconds. The measurement these channels is hampered by various technical constraints, including the small size and extended electrotonic structure of HPCs and the diverse preparations used in experiments. Modelling the electrical behavior of HPCs with computer simulations is one method of integrating data from a variety of sources in order to develop a consistent description for this cell type.

In the model referred to here putative mechanisms for voltage-dependent and calcium-dependent channel gating have been used to generate simulations of the somatic electrical behavior of HPCs, and to suggest mechanisms for information processing at the single cell level. The model has also been used to suggest experimental protocols designed to test the validity of simulation results. Model simulations qualitatively or quantitatively reproduce a wide range of somatic electrical behavior in HPCs, and explicitly demonstrate possible functional roles for the various currents [1].

The model presently includes descriptions of eleven non-linear somatic currents, including three putative $Na^+$ currents – $I_{Na-trig}$, $I_{Na-rep}$, and $I_{Na-tail}$; six $K^+$ currents that have been reported in the literature – $I_{DR}$ (Delayed Rectifier), $I_A$, $I_C$, $I_{AHP}$ (After-hyperpolarization), $I_M$, and $I_Q$; and two $Ca^{2+}$ currents, also reported previously – $I_{Ca}$ and $I_{CaS}$.

The electrotonic structure of the HPC is modelled with a soma/short-cable approximation, and the dendrites are assumed to be linear. While the conditions for reducing the dendritic tree to a single cable are not met for HPC (the so-called Rall conditions [3]), the $Z_{in}$ of the cable is close to that of the tree. In addition, although HPC dendrites have non-linear membrane, it assumed that as a first approximation the contribution of currents from this membrane may be ignored in the somatic response to somatic stimulus. Likewise, the model structure assumes that axon-soma current under these conditions can be lumped into the soma circuit.

In part this paper will address the following question: if neural nets are realizable using elements that have simple integrative all-or-nothing responses, connected to each other with regenerative conductors, then what is the function for all the channels observed experimentally in real neurons? The results of this HPC model study suggest some purpose for these complexities, and in this paper we shall investigate some of the possible roles of non-linear channels in neuronal information processing. However, given the speculative nature of many of the currents that we have presented in the model, it is important to view results based on the interaction of the many model elements as preliminary.

## 3    Defining Neural Information Coding is the First Step in Describing Biological Computations

Determination of computational properties of neurons requires *a priori* assumptions as to how information is encoded in neuronal output. The classical description assumes that information is encoded as spike frequency. However, a single output variable, proportional to firing frequency, ignores other potentially information-rich degrees of freedom, including:

- Relative phase of concurrent inputs.

- Frequency modulation during single bursts.

- Cessation of firing due to intrinsic mechanisms.

- Spike shape.

Note that these variables apply to patterns of repetitive firing[1]. The relative phase of different inputs to a single cell is very important at low firing rates, but becomes less so as firing frequency approaches the time constant of the postsynaptic membrane or some other rate-limiting process in the synaptic transduction (e.g. neurotransmitter release or post synaptic channel activation/deactivation kinetics). Frequency modulation during bursts/spike trains may be important in the interaction of a given axon's output with other inputs at the target neuron. Cessation of firing due to mechanisms intrinsic to the cell (as opposed to the end of input) may be

important, for example, in that cell's transmission function. Finally, modulation of spike shape may have several consequences, which will be discussed later.

# 4    Physiological Modulation of HPC Currents

In order for modulation of HPC currents to be considered as potential information processing mechanisms *in vivo*, it is necessary to identify physiological modulators. For several of the currents described here such factors have been identified. For example, there is evidence that $I_M$ is inhibited by muscarinic (physiologically, cholinergic) agonists [2], that $I_A$ is inhibited by acetylcholine [6], and that $I_{AHP}$ is inhibited by noradrenaline [5]. In fact, the list of neurotransmitters which are active non-synaptically is growing rapidly. It remains to be seen whether there are as yet undiscovered mechanisms for modulating other HPC currents, for example the three $Na^+$ currents proposed in the present model. Some possible consequences of such mechanisms will be discussed later.

# 5    HPC Currents and Information Processing

The role of a given channel on the HPC electrical response depends on its temporal characteristics as a function of voltage, intracellular messengers, and other variables. This is complicated by the fact that the opening and closing of channels is equivalent to varying *conductances*, allowing both linear and non-linear operations (e.g. [4] and [7]). In particular, a current which is activated/deactivated over a period of hundreds of milliseconds will, to a first approximation, act by slowly changing the time constant of the membrane. At the other extreme, currents which activate/deactivate with sub-millisecond time constants act by changing the trajectory of the membrane voltage in complicated ways. The classic example of this is the role of $Na^+$ currents underlying the action potential.

To investigate how the different HPC currents may contribute to the information processing of this neuron, we have looked at how each current shapes the HPC response to a simple repertoire of inputs. At this stage in our research the inputs have been very basic – short somatic current steps that evoke single spikes, long lasting somatic current steps that evoke spike trains, and current steps at the distal end of the dendritic cable. By examining the response to these inputs the functional roles of the HPC

| Current | Spike Shape | Spike Threshold | $\tau_m$/Frequency-Intensity |
|---------|-------------|-----------------|------------------------------|
| $I_{Na-trig}$ | + | +++ | − |
| $I_{Na-rep}$ | + | ++ | +++ |
| $I_{Ca}$ | − (++) | − (+) | + (+++) |
| $I_{DR}$ | ++ | + | ++ |
| $I_A$ | + | ++ | ++ |
| $I_C$ | + | − | +++ |
| $I_{AHP}$ | − | ++ | +++ |
| $I_M$ | − | + | + |

Table 1: Putative functional roles of HPC somatic currents. Entries in parentheses indicate secondary role, e.g. $Ca^{2+}$ activation of $K^+$ current.

currents can be tentatively grouped into three (non-exclusive) categories:

- Modulation of spike shape.

- Modulation of firing threshold, both for single and repetitive spikes.

- Modulation of semi-steady-state membrane time constant.

- Modulation of repetitive firing, specifically the relationship between strength of tonic input and frequency of initial burst and later "steady state" spike train.

Table 1 summarizes speculative roles for some of the HPC currents as suggested by the simulations. Note that while all four of the listed characteristics are interrelated, the last two are particularly so and are lumped together in Table 1.

## 5.1 Possible Roles for Modulation of FI Characteristic

Again, it has been traditionally assumed that neural information is encoded by (steady-state) frequency modulation, e.g. the number of spikes per second over some time period encodes the output information of a neuron. For example, muscle fiber contraction is approximately proportional to the spike frequency of its motor neuron [2]. If the physiological inhibition of a specific

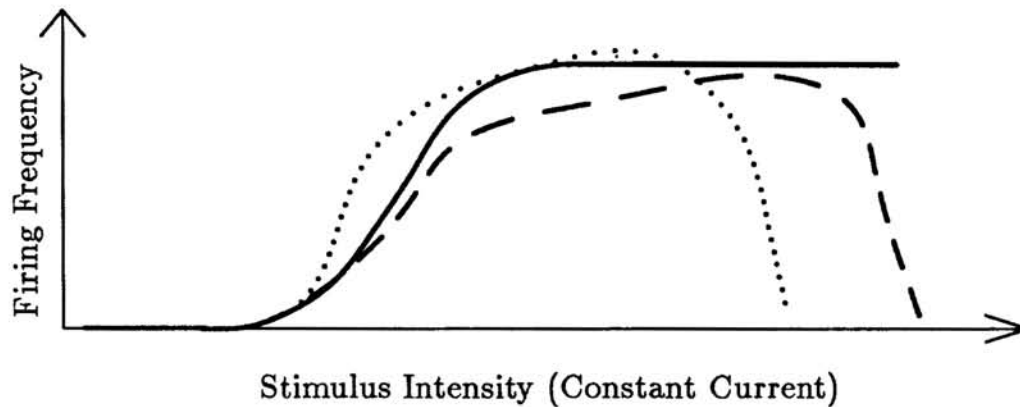

Figure 1: Classical relation between total neuronal input (typically tonic current stimulus) and spike firing frequency [solid line] and (qualitative) biological relationships [dashed and dotted lines]. The dotted line applies when $I_{Na-rep}$ is blocked.

current changes the FI characteristic, this allows one way to modulate that neuron's information processing by various agents.

Figure 1 contrasts the classical input-output relation of a neuron and more biological input-output relations. The relationships have several features which can be potentially modulated either physiologically or pathologically, including saturation, threshold, and shape of the curves. Note in particular the cessation of output with increased stimulation, as the depolarizing stimulus prevents the resetting of the transient inward currents.

For the HPC, simulations show (Figure 2 and Figure 3) that blocking the putative $I_{Na-rep}$ has the effect of causing the cell to "latch-up" in response to tonic stimulus that would otherwise elicit stable spike trains. Both depolarizing currents and repolarizing currents play a role here. First, spike upstroke is mediated by both $I_{Na-rep}$ and the lower threshold $I_{Na-trig}$; at high stimuli repolarization between spikes does not get low enough to reset $I_{Na-trig}$. Second, spikes due to only one of these $Na^+$ currents are weaker and as a result do not activate the repolarizing $K^+$ currents as much as normal because a) reduced time at depolarized levels activates the voltage-dependent $K^+$ currents less and b) less $Ca^{2+}$ influx with smaller spikes reduces the $Ca^{2+}$-dependent activation of some $K^+$ currents. The net result is that repolarization between spikes is weaker and, again, does not reset $I_{Na-trig}$.

Although the current being modulated here ($I_{Na-rep}$) is theoretical, the

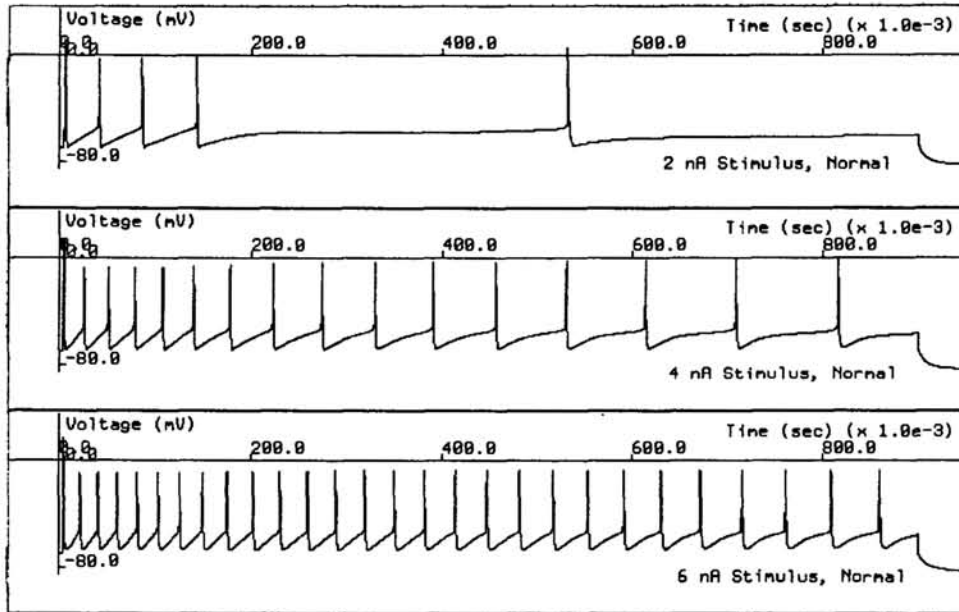

Figure 2: Simulation of repetitive firing in response to constant current injection into the soma. In this series, with the "normal" cell, a stimulus of about 8 nA (not shown) will cause to cell to fire a short burst and then cease firing.

possibility of selective blocking of $I_{Na-rep}$ allows a mechanism for shifting the saturation of the neuron's response to the left and, as can be seen by comparing Figures 2 and 3, making the FI curve steeper over the response range.

## 5.2 Possible Roles for Modulation of Spike Threshold

The somatic firing threshold determines the minimal input for eliciting a spike, and in effect change the sensitivity of a cell. As a simple example, blocking $I_{Na-trig}$ in the HPC model raises threshold by about 10 millivolts. This could cause the cell to ignore input patterns that would otherwise generate action potentials.

There are two aspects of the firing "threshold" for a cell – static and dynamic. Thus, the *rate* at which the soma membrane approaches threshold is important along with the magnitude of that threshold. In general the threshold level rises with a slower depolarization for several reasons, including partial inactivation of inward currents (e.g. $I_{Na-trig}$) and partial activation of outward currents (e.g. $I_A$ [8]) at subthreshold levels.

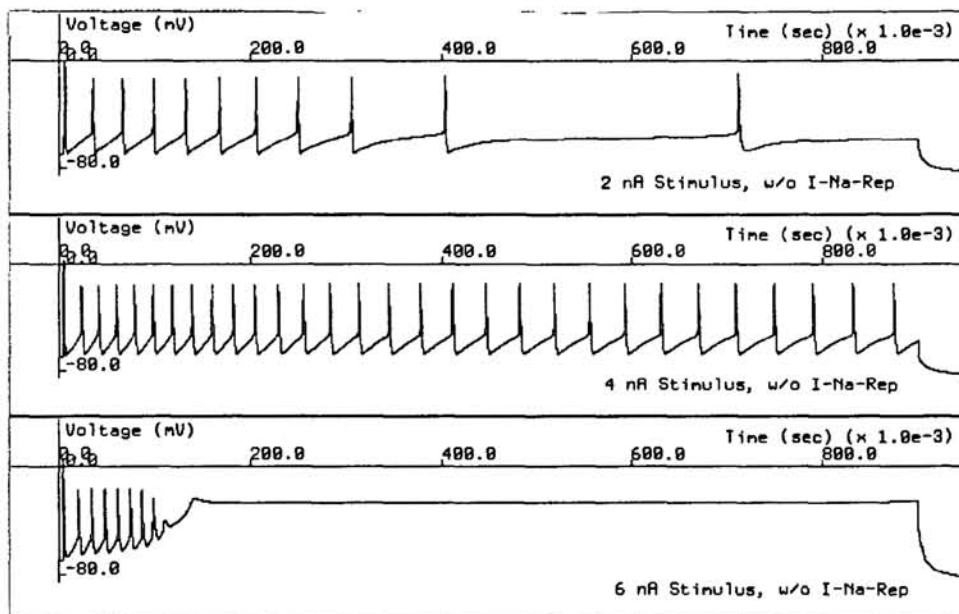

Figure 3: Blocking one of the putative $Na^+$ currents ($I_{Na-rep}$) causes the HPC repetitive firing response to fail at lower stimulus than "normal". This corresponds to the leftward shift in the saturation of the response curve shown in Figure 1.

Thus it is possible, for example, that $I_A$ helps to distinguish tonic dendritic distal synaptic input from proximal input. For input that eventually will supply the same depolarizing current at the soma, dendritic input will have a slower onset due to the cable properties of the dendrites. This slow onset could allow $I_A$ to delay the onset of the spike or spikes. A similar depolarizing current applied more proximally would have a faster onset. Sub-threshold activation of $I_A$ on the depolarizing phase would then be insufficient to delay the spike.

## 5.3 Possible Roles for Modulation of Somatic Spike Shape

How important is the *shape* of an individual spike generated at the soma? First, we can assume that spike shape, in particular spike width, is unimportant at the soma spike-generating membrane – once the soma fires, it fires. However, the effect of the spike beyond the soma may or may not depend on the spike shape, and this is dependent on both the degree which spike propagation is linear and on the properties of the pre-synaptic membrane.

Axon transmission is both a linear and non-linear phenomena, and the shorter the axon's electrotonic length, the more the shape of the somatic

action potential will be preserved at the distal pre-synaptic terminal. At one extreme, an axon could transmit the spike a purely non-linear fashion – once threshold was reached, the classic "all-or-nothing" response would transmit a stereotyped action potential whose shape would be independent of the post-threshold soma response. At the other extreme, i.e. if the axonal membrane were purely linear, the propagation of the somatic event at any point down the axon would be a linear convolution of the somatic signal and the axon cable properties. It is likely that the situation in the brain lies somewhere between these limits, and will depend on the wavelength of the spike, the axon non-linearities and the axon length.

What role could be served by the somatic action potential shape modulating the pre-synaptic terminal signal? There are at least three possibilities. First, it has been demonstrated that the release of transmitter at some pre-synaptic terminals is not an "all-or-nothing" event, and in fact is a function of the pre-synaptic membrane voltage waveform. Thus, modulation of the somatic spike width may determine how much transmitter is released down the line, providing a mechanism for changing the effective strength of the spike as seen by the target neuron. Modulation of somatic spike width could be equivalent to a modulation of the "loudness" of a given neuron's message.

Second, pyramidal cell axons often project collateral branches back to the originating soma, forming axo-somatic synapses which result in a feedback loop. In this case, modulation of the somatic spike could affect this feedback in complicated ways, particularly since the collaterals are typically short.

Finally, somatic spike shape may also play a role in the transmission of spikes at axonal branch points. For example, consider a axonal branch point with an impedance mismatch and two daughter branches, one thin and one thick. Here a spike that is too narrow may not be able to depolarize the thick branch sufficiently for transmission of the spike down that branch, with the spike propagating only down the thin branch. Conversely, a wider spike may be passed by both branches. Modulation of the somatic spike shape could then be used to direct how a cell's output is broadcast, some times allowing transmission to all the destinations of an HPC, and at other times inhibiting transmission to a limited set of the target neurons.

For HPCs much evidence has been obtained which implicate the roles of various HPC currents on modulating somatic spike shape, for example the $Ca^{2+}$-dependent $K^+$ current $I_C$ [9]. Simulations which demonstrate the effect of $I_C$ on the shape of individual action potentials are shown in Figure 4.

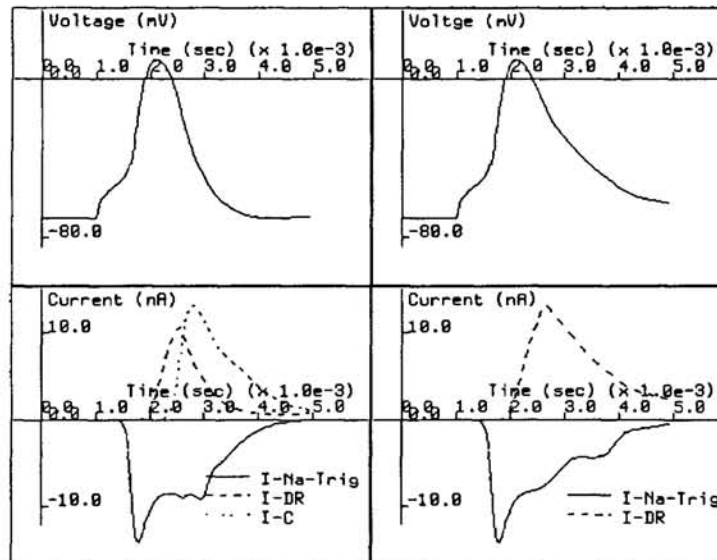

Figure 4: Role of $I_C$ during repolarization of spike. In the simulation on the left, $I_C$ is the largest repolarizing current. In the simulation on the right, blocking $I_C$ results in an wider spike.

# 6   The Assumption of Somatic Vs. Non-Somatic Currents

In this research the *somatic* response of the HPC has been modelled under the assumption that the data on HPC currents reflect activity of channels localized at the soma. However, it must be considered that all channel proteins, regardless of their final functional destination, are manufactured at the soma. Some of the so-called somatic channels may therefore be vestiges of channels intended for dendritic, axonal, or pre-synaptic membrane. For example, if the spike-shaping channels are intended to be expressed for pre-synaptic membrane, then modulation of these channels by endogenous factors (e.g. ACh) takes place at target neuron. This may seem disadvantageous if a factor is to act selectively on some afferent tract. On the other hand, in the dendritic field of a given neuron it is possible only some afferents have certain channels, thus allowing selective response to modulating agents. These possibilities further expand the potential roles of membrane channels for computation.

# 7 Other Possible Roles of Currents for Modulating HPC Response

There are many other potential ways that HPC currents may modulate the HPC response. For example, the relationship between intracellular $Ca^{2+}$ and the $Ca^{2+}$-dependent $K^+$ currents, $I_C$ and $I_{AHP}$, may indicate possible information processing mechanisms.

Intracellular $Ca^{2+}$ is an important second messenger for several intracellular processes, for example muscular contraction, but excessive $[Ca^{2+}]_{in}$ is noxious. There are at least three negative feedback mechanisms for limiting the flow of $Ca^{2+}$ : voltage-dependent inactivation of $Ca^{2+}$ currents; reduction of $E_{Ca}$ (and thus the $Ca^{2+}$ driving force) with $Ca^{2+}$ influx; and the just mentioned $Ca^{2+}$-mediation of repolarizing currents. A possible information processing mechanism could be by modulation of $I_{AHP}$, which plays an important role in limiting repetitive firing[3]. Simulations suggest that blocking this current causes $I_C$ to step in and eventually limit further repetitive firing, though after many more spikes in a train. Blocking both these currents may allow other mechanisms to control repetitive firing, perhaps ones that operate independently of $[Ca^{2+}]_{in}$. Conceivably, this could put the neuron into quite a different operating region.

# 8 Populations of Neurons Vs. Single Cells: Implications for Graded Modulation of HPC Currents

In this paper we have considered the all-or-nothing contribution of the various channels, i.e. the entire population of a given channel type is either activated normally or all the channels are disabled/blocked. This description may be oversimplified in two ways. First, it is possible that a blocking mechanism for a given channel may have a graded effect. For example, it is possible that cholinergic input is not homogeneous over the soma membrane, or that at a given time only a portion of these afferents are activated. In either case it is possible that only a portion of the cholinergic receptors are bound, thus inhibiting a portion of channels. Second, the result of channel inhibition by neuromodulatory projections must consider both single cell

response and population response, the size of the population depending on the neuro-architecture of a cortical region and the afferents. For example, activation of a cholinergic tract which terminates in a localized hippocampal region may effect thousands of HPCs. Assuming that the $I_M$ of individual HPCs in the region may be either turned on or off completely with some probability, the behavior of the *population* will be that of a graded response of $I_M$ inhibition. This graded response will in turn depend on the strength of the cholinergic tract activity.

The key point is that the information processing properties of isolated neurons may be reflected in the behavior of a population, and vica-versa. While it is likely that removal of a single pyramidal cell from the hippocampus will have zero functional effect, no neuron is an island. Understanding the central nervous system begins with the spectrum of behavior in its functional units, which may range from single channels, to specific areas of a dendritic tree, to the single cell, to cortical or nuclear subfields, on up through the main subsystems of CNS.

## Footnotes

[1]Single spikes may be considered as degenerate cases of repetitive firing responses.

[2]In fact, where action potential propagation is a stereotyped phenomena, such as in long axons, then the timing of spikes is the only parameter that may be modulated.

[3]The slowing down of the spike trains in Figure 2 and Figure 3 is mainly due to the buildup of $[Ca^{2+}]_{in}$, which progressively activates more $I_{AHP}$.

# References

[1] L. Borg-Graham. *Modelling the Somatic Electrical Behavior of Hippocampal Pyramidal Neurons.* Master's thesis, Massachusetts Institute of Technology, 1987.

[2] J. Halliwell and P. Adams. Voltage clamp analysis of muscarinic excitation in hippocampal neurons. *Brain Research*, 250:71–92, 1982.

[3] J. J. B. Jack, D. Noble, and R. W. Tsien. *Electric Current Flow In Excitable Cells.* Clarendon Press, Oxford, 1983.

[4] C. Koch and T. Poggio. Biophysics of computation: neurons, synapses and membranes. *C.B.I.P. Paper*, (008), 1984. Center for Biological Information Processing, MIT.

[5] D. Madison and R. Nicoll. Noradrenaline blocks accommodation of pyramidal cell discharge in the hippocampus. *Nature*, 299:, Oct 1982.

[6] Y. Nakajuma, S. Nakajima, R. Leonard, and K. Yamaguchi. Actetylcholine inhibits a-current in dissociated cultured hippocampal neurons. *Biophysical Journal*, 49:575a, 1986.

[7] T. Poggio and V. Torre. *Theoretical Approaches to Complex Systems, Lecture Notes in Biomathematics*, pages 28–38. Volume 21, Springer Verlag, Berlin, 1978. A New Approach to Synaptic Interaction.

[8] J. Storm. A-current and ca-dependent transient outward current control the initial repetitive firing in hippocampal neurons. *Biophysical Journal*, 49:369a, 1986.

[9] J. Storm. Mechanisms of action potential repolarization and a fast after-hyperpolarization in rat hippocampal pyramidal cells. *Journal of Physiology*, 1986.